# Learning from Candidate Labeling Sets

**Luo Jie**
Idiap Research Institute and EPF Lausanne
jluo@idiap.ch

**Francesco Orabona**
DSI, Università degli Studi di Milano
orabona@dsi.unimi.it

## Abstract

In many real world applications we do not have access to fully-labeled training data, but only to a list of *possible* labels. This is the case, e.g., when learning visual classifiers from images downloaded from the web, using just their text captions or tags as learning oracles. In general, these problems can be very difficult. However most of the time there exist different implicit sources of information, coming from the relations between instances and labels, which are usually dismissed. In this paper, we propose a semi-supervised framework to model this kind of problems. Each training sample is a bag containing multi-instances, associated with a set of candidate labeling vectors. Each labeling vector encodes the possible labels for the instances in the bag, with only one being fully correct. The use of the labeling vectors provides a principled way not to exclude any information. We propose a large margin discriminative formulation, and an efficient algorithm to solve it. Experiments conducted on artificial datasets and a real-world images and captions dataset show that our approach achieves performance comparable to an SVM trained with the ground-truth labels, and outperforms other baselines.

## 1 Introduction

In standard supervised learning, each training sample is associated with a label, and the classifier is usually trained through the minimization of the empirical risk on the training set. However, in many real world problems we are not always so lucky. Partial data, noise, missing labels and other similar common issues can make you deviate from this ideal situation, moving the learning scenario from supervised learning to semi-supervised learning [7, 26].

In this paper, we investigate a special kind of semi-supervised learning which considers ambiguous labels. In particular each training example is associated with several *possible* labels, among which only one is correct. Intuitively this problem can be arbitrarily hard in the worst case scenario. Consider the case when one noisy label is consistently appearing together with the true label: in this situation we could not tell them apart. Despite that, learning could still be possible in many typical real world scenarios. Moreover, in real problems samples are often gathered in groups, and the intrinsic nature of the problem could be used to constrain the possible labels for the samples from the same group. For example, we might have that two labels can not appear together in the same group or a label can appear only once in each group, as, for example, a specific face in an image.

Inspired by these scenarios, we focus on the general case where we have *bags* of instances, with each bag associated with a set of several possible labeling vectors, and among them only one is fully correct. Each labeling vector consists of labels for each corresponding instance in the bag. For easy reference, we call this type of learning problem a Candidate Labeling Set (CLS) problem.

As labeled data is usually expensive and hard to obtain, CLS problems naturally arise in many real world tasks. For example, in computer vision and information retrieval domains, photographs collections with tags have motivated the studies on learning from weakly annotated images [2], as each image (bag) can be naturally partitioned into several patches (instances), and one could assume that each tag should be associated with at least one patch. High-level knowledge, such as spatial

correlations (e.g. "sun in sky" and "car on street"), have been explored to prune down the labeling possibilities [14]. Another similar task is to learn a face recognition system from images gathered from news websites or videos, using the associated text captions and video scripts [3, 8, 16, 13]. These works use different approaches to integrate the constraints, such as that two faces in one image could not be associated with the same name [3], mouth motion and gender of the person [8], or modeling both names and action verbs jointly [16]. Another problem is the multiple annotators scenario, where each data is associated with the labels given by independently hired annotators. The annotators can disagree on the data and the aim is to recover the true label of each sample. All these problems can be naturally casted into the CLS framework.

The contribution of this paper is a new formal way to cast the CLS setup into a learning problem. We also propose a large margin formulation and an efficient algorithm to solve it. The proposed Maximum Margin Set learning (MMS) algorithm, can scale to datasets of the order of $10^5$ instances, reaching performances comparable to fully-supervised learning algorithms.

**Related works.**  This type of learning problem dates back to the work of Grandvalet in [12]. Later Jin and Ghaharmani [17] formalized it and proposed a general framework for discriminative models. Our work is also closely related to the ambiguous labeling problem presented in [8, 15]. Our framework generalizes them, to the cases where instances and possible labels come in the form of bags. This particular generalization gives us a principled way for using different kinds of prior knowledge on instances and labels correlation, without hacking the learning algorithm. More specifically, prior knowledge, such as pairwise constraints [21] and mutual exclusiveness of some labels, can be easily encoded in the labeling vectors. Although several works have focused on integrating these weakly labeled information that are complementary to the labeled or unlabeled training data into existing algorithms, these approaches are usually computational expensive. On the other hand, in our framework we have the opposite behavior: the more prior knowledge we exploit to construct the candidate set, the better the performance and the faster the algorithm will be.

Other lines of research which are related to this paper are multiple-instance learning (MIL) problems [1, 5, 10], and multi-instance multi-label learning (MIML) problems [24, 25] which extends the binary MIL setup to multi-labels scenario. In both setups, several instances are grouped into bags, and their labels are not individually given but assigned to the bags directly. However, contrary to our framework, in MIML noisy labeling is not allowed. In other words, all the labels being assigned to the bags are assumed to be true. Moreover, current MIL and MIML algorithms usually rely on a 'key' instance in the bag [1] or they transform each bag into single instance representation [25], while our algorithm makes an explicit effort to label every instance in a bag and to consider all of them during learning. Hence, it has a clear advantage in problems where the bags are dense in labeled instances and instances in the same bag are independent, as opposed to the cases when several instances jointly represent a label. Our algorithm is also related to Latent Structural SVMs [22], where the correct labels could be considered as latent variables.

## 2   Learning from Candidate Labeling Sets

**Preliminaries.**  In this section, we formalize the CLS setting, which is a generalization of the ambiguous labeling problem described in [17] from single instances to bags of instances.

In the following we denote vectors by bold letters, e.g. $\boldsymbol{w}, \boldsymbol{y}$, and use calligraphic font for sets, e.g., $\mathcal{X}$. In the CLS setting, the $N$ training data are provided in the form $\{\mathcal{X}_i, \mathcal{Z}_i\}_{i=1}^N$, where $\mathcal{X}_i$ is a bag of $M_i$ instances, $\mathcal{X}_i = \{\boldsymbol{x}_{i,m}\}_{m=1}^{M_i}$, and $\boldsymbol{x}_{i,m} \in \mathbb{R}^d, \forall\, i = 1, \dots, N, \ \ m = 1, \dots, M_i$. The associated set of $L_i$ candidate labeling vectors is $\mathcal{Z}_i = \{\boldsymbol{z}_{i,l}\}_{l=1}^{L_i}$, where $\boldsymbol{z}_{i,l} \in \mathcal{Y}^{M_i}$, and $\mathcal{Y} = \{1, ..., C\}$. In other words there are $L_i$ different combinations of $M_i$ labels for the $M_i$ instances in the $i$-th bag. We assume that the correct labeling vector for $\mathcal{X}_i$ is present in $\mathcal{Z}_i$, while the other labeling vectors maybe partially correct or even completely wrong. It is important to point out that this assumption is not equivalent to just associating $L_i$ candidate labels to each instance. In fact, in this way we also encode explicitly the correlations between instances and their labels in a bag. For example, consider a two instances bag $\{\boldsymbol{x}_{i,1}, \boldsymbol{x}_{i,2}\}$: if it is known that they can only come from classes 1 and 2, and they can not share the same label, then $\boldsymbol{z}_{i,1} = [1, 2], \boldsymbol{z}_{i,2} = [2, 1]$ will be the candidate labeling vectors for this bag, while the other possibilities are excluded from the labeling set. In the following we will assume that the labeling set $\mathcal{Z}_i$ is given with the training set. In Section 4.2 we will give a practical example on how to construct this set using the prior knowledge on the task.

Given the training data $\{\mathcal{X}_i, \mathcal{Z}_i\}_{i=1}^N$, we want to learn a function $f(\boldsymbol{x})$, to correctly predict the class of each single instance $\boldsymbol{x}$, coming from the same distribution. The problem would become the standard multiclass supervised learning if there is only one labeling vector in every labeling set $\mathcal{Z}_i$, i.e. $L_i = 1$. On the other hand, given a set of $C$ labels, without any prior knowledge, a bag of $M_i$ instances could have maximum $C^{M_i}$ labeling vectors, which becomes a clustering problem. However, we are more interested in situations when $L_i \ll C^{M_i}$.

## 2.1 Large-margin formulation

We introduce here a large margin formulation to solve the CLS problem. It is helpful to first define by $\mathcal{X}$ the generic bag of $M$ instances $\{\boldsymbol{x}_1, \ldots \boldsymbol{x}_M\}$, $\mathcal{Z} = \{\boldsymbol{z}_1, \ldots, \boldsymbol{z}_L\}$ the generic set of candidate labeling vectors, and $\boldsymbol{y} = \{y_1, \ldots, y_M\}, \boldsymbol{z} = \{z_1, \ldots, z_M\} \in \mathcal{Y}^M$ two labeling vectors.

We start by introducing the loss function that assumes the true label $y_m$ of each instance $\boldsymbol{x}_m$ is known

$$\ell_\Delta(\boldsymbol{z}, \boldsymbol{y}) = \sum_{m=1}^M \Delta(z_m, y_m) \, , \tag{1}$$

where $\Delta(z_m, y_m)$ is a non-negative loss function measuring how much we pay for having predicted $z_m$ instead of $y_m$. For example $\Delta(z_m, y_m)$ can be defined as $\mathbf{1}(z_m \neq y_m)$, where $\mathbf{1}$ is the indicator function. Hence, if the vector $\boldsymbol{z}$ is the predicted label for the bag, $\ell_\Delta(\boldsymbol{z}, \boldsymbol{y})$ simply counts the number of misclassified instances in the bag.

However, the true labels are unknown, and we only have access to the set $\mathcal{Z}$, knowing that the true labeling vector is in $\mathcal{Z}$. So we use a proxy of this loss function, and propose the ambiguous version of this loss:

$$\ell_\Delta^A(\boldsymbol{z}, \mathcal{Z}) = \min_{\boldsymbol{z}' \in \mathcal{Z}} \ell_\Delta(\boldsymbol{z}, \boldsymbol{z}') \, .$$

We also define, with a small abuse of notation, $\ell_\Delta^A(\mathcal{X}, \mathcal{Z}; f) = \ell_\Delta^A(f(\mathcal{X}), \mathcal{Z})$, where $f(\mathcal{X})$ returns a labeling vector which consists of labels for each instance in the bag $\mathcal{X}$. It is obvious that this loss underestimates the true loss. Nevertheless, we can easily extend [8, Proposition 3.1 to 3.3] to the bag case, and prove that $\ell_\Delta^A/(1 - \eta)$ is an upper bound to $\ell_\Delta$ in expectation, where $\eta$ is a factor between 0 and 1, and its value depends on the hardness of the problem. Like the definition in [8], $\eta$ corresponds to the maximum probability of an extra label co-occurring with the true label over all labels and instances. Hence, minimizing the ambiguous loss we are actually minimizing an upper bound of the true loss. It is a known problem that direct minimization of this loss is hard, so in the following we introduce another loss that upper bounds $\ell_\Delta^A$ which can be minimized efficiently.

We assume that the prediction function $f(\boldsymbol{x})$ we are searching for is equal to $\arg\max_{y \in \mathcal{Y}} F(\boldsymbol{x}, y)$. In this framework we can interpret the value of $F(\boldsymbol{x}, y)$ as the confidence of the classifier in assigning $\boldsymbol{x}$ to the class $y$. We also assume the standard linear model used in supervised multiclass learning [9]. In particular the function $F(\boldsymbol{x}, y)$ is set to be $\boldsymbol{w} \cdot \phi(\boldsymbol{x}) \otimes \psi(y)$, where $\phi$ and $\psi$ are the feature and label space mapping [20], and $\otimes$ is the Kronecker product[1]. We can now define $\mathbf{F}(\mathcal{X}, \boldsymbol{y}; \boldsymbol{w}) = \sum_{m=1}^M F(\boldsymbol{x}_m, y_m)$, which intuitively is gathering from each instance in $\mathcal{X}$ the confidence on the labels in $\boldsymbol{y}$. With the definitions above, we can rewrite the function $\mathbf{F}$ as

$$\mathbf{F}(\mathcal{X}, \boldsymbol{y}; \boldsymbol{w}) = \sum_{m=1}^M F(\boldsymbol{x}_m, y_m) = \sum_{m=1}^M \boldsymbol{w} \cdot \phi(\boldsymbol{x}_m) \otimes \psi(y_m) = \boldsymbol{w} \cdot \Phi(\mathcal{X}, \boldsymbol{y}) \, , \tag{2}$$

where we defined $\Phi(\mathcal{X}, \boldsymbol{y}) = \sum_{m=1}^M \phi(\boldsymbol{x}_m) \otimes \psi(y_m)$. Hence the function $\mathbf{F}$ can be defined as the scalar product between $\boldsymbol{w}$ and a joint feature map between the bag $\mathcal{X}$ and the labeling vector $\boldsymbol{y}$.

**Remark.** *If the prior probabilities of every candidate labeling vectors $\boldsymbol{z}_l \in \mathcal{Z}$ are also available, they could be incorporated by slightly modifying the feature mapping scheme in (2).*

We can now introduce the following loss function

$$\ell_{max}(\mathcal{X}, \mathcal{Z}; \boldsymbol{w}) = \left| \max_{\bar{\boldsymbol{z}} \notin \mathcal{Z}} \left( \ell_\Delta^A(\bar{\boldsymbol{z}}, \mathcal{Z}) + \mathbf{F}(\mathcal{X}, \bar{\boldsymbol{z}}; \boldsymbol{w}) \right) - \max_{\boldsymbol{z} \in \mathcal{Z}} \mathbf{F}(\mathcal{X}, \boldsymbol{z}; \boldsymbol{w}) \right|_+ \tag{3}$$

where $|x|_+ = \max(0, x)$. The following proposition shows that $\ell_{max}$ upper bounds $\ell_\Delta^A$.

**Proposition.** $\ell_{max}\left(\mathcal{X}, \mathcal{Z}; \boldsymbol{w}\right) \geq \ell_{\Delta}^{A}\left(\mathcal{X}, \mathcal{Z}; \boldsymbol{w}\right)$ .

*Proof.* Define $\hat{\boldsymbol{z}} = \arg\max_{\boldsymbol{z} \in \mathcal{Y}^M} \mathbf{F}(\mathcal{X}, \boldsymbol{z}; \boldsymbol{w})$. If $\hat{\boldsymbol{z}} \in \mathcal{Z}$ then $\ell_{max}\left(\mathcal{X}, \mathcal{Z}; \boldsymbol{w}\right) \geq \ell_{\Delta}^{A}\left(\mathcal{X}, \mathcal{Z}; \boldsymbol{w}\right) = 0$. We now consider the case in which $\hat{\boldsymbol{z}} \notin \mathcal{Z}$. We have that

$$\ell_{\Delta}^{A}\left(\mathcal{X}, \mathcal{Z}; \boldsymbol{w}\right) \leq \ell_{\Delta}^{A}(\hat{\boldsymbol{z}}, \mathcal{Z}) + \mathbf{F}(\mathcal{X}, \hat{\boldsymbol{z}}; \boldsymbol{w}) - \max_{\boldsymbol{z} \in \mathcal{Z}} \mathbf{F}(\mathcal{X}, \boldsymbol{z}; \boldsymbol{w})$$

$$\leq \max_{\bar{\boldsymbol{z}} \notin \mathcal{Z}} \left(\ell_{\Delta}^{A}(\bar{\boldsymbol{z}}, \mathcal{Z}) + \mathbf{F}(\mathcal{X}, \bar{\boldsymbol{z}}; \boldsymbol{w})\right) - \max_{\boldsymbol{z} \in \mathcal{Z}} \mathbf{F}(\mathcal{X}, \boldsymbol{z}; \boldsymbol{w}) \leq \ell_{max}\left(\mathcal{X}, \mathcal{Z}; \boldsymbol{w}\right) . \qquad \square$$

The loss $\ell_{max}$ is non-convex, due to the second $\max(\cdot)$ function inside, but in Section 3 we will introduce an algorithm to minimize it efficiently.

## 2.2 A probabilistic interpretation

It is possible to gain additional intuition on the proposed loss function $\ell_{max}$ through a probabilistic interpretation of the problem. It is helpful to look at the discriminative model for supervised learning first, where the goal is to learn the model parameters $\theta$ for the function $P(y|\boldsymbol{x}; \theta)$, from a pre-defined modeling class $\Theta$. Instead of directly maximizing the log-likelihood for the training data, an alternative way is to maximize the log-likelihood ratio between the correct label and the most likely incorrect one [9]. On the other hand, in the CLS setting the correct labeling vector for $\mathcal{X}$ is unknown, but it is known to be a member of the candidate set $\mathcal{Z}$. Hence we could maximize the log-likelihood ratio between $P(\mathcal{Z}|\mathcal{X}; \theta)$ and the most likely incorrect labeling vector which is not member of $\mathcal{Z}$ (denoted as $\bar{\boldsymbol{z}}$). However, the correlations between different vectors in $\mathcal{Z}$ are not known, so the inference could be arbitrarily hard. Instead, we could approximate the problem by considering just the most likely correct member of $\mathcal{Z}$. It can be easily verified that $\max_{\boldsymbol{z} \in \mathcal{Z}} P(\boldsymbol{z}|\mathcal{X}; \theta)$ is a lower bound of $P(\mathcal{Z}|\mathcal{X}; \theta)$. The learning problem becomes to minimize the ratio for the bag:

$$-\log \frac{P(\mathcal{Z}|\mathcal{X}; \theta)}{\max_{\bar{\boldsymbol{z}} \notin \mathcal{Z}} P(\bar{\boldsymbol{z}}|\mathcal{X}; \theta)} \approx -\log \frac{\max_{\boldsymbol{z} \in \mathcal{Z}} P(\boldsymbol{z}|\mathcal{X}; \theta)}{\max_{\bar{\boldsymbol{z}} \notin \mathcal{Z}} P(\bar{\boldsymbol{z}}|\mathcal{X}; \theta)} . \tag{4}$$

If we assume independence between the instances in the bag, (4) can be factorized as:

$$-\log \frac{\max_{\boldsymbol{z} \in \mathcal{Z}} \prod_m P(z_m|\boldsymbol{x}_m; \theta)}{\max_{\bar{\boldsymbol{z}} \notin \mathcal{Z}} \prod_m P(\bar{z}_m|\boldsymbol{x}_m; \theta)} = \max_{\bar{\boldsymbol{z}} \notin \mathcal{Z}} \sum_m \log P(\bar{z}_m|\boldsymbol{x}_m; \theta) - \max_{\boldsymbol{z} \in \mathcal{Z}} \sum_m \log P(z_m|\boldsymbol{x}_m; \theta) .$$

If we take the margin into account, and assume a linear model for the log-posterior-likelihood, we obtain the loss function in (3).

## 3 MMS: The Maximum Margin Set Learning Algorithm

Using the square norm regularizer as in the SVM and the loss function in (3), we have the following optimization problem for the CLS learning problem:

$$\min_{\boldsymbol{w}} \frac{\lambda}{2}\|\boldsymbol{w}\|_2^2 + \frac{1}{N} \sum_{i=1}^{N} \ell_{max}\left(\mathcal{X}_i, \mathcal{Z}_i; \boldsymbol{w}\right) \tag{5}$$

This optimization problem (5) is non-convex due to the non-convex loss function (3). To convexify this problem, one could approximate the second $\max(\cdot)$ in (3) with the average over all the labeling vectors in $\mathcal{Z}_i$. Similar strategies have been used in several analogous problems [8, 24]. However, the approximation could be very loose if the number of labeling vectors is large. Fortunately, although the loss function is not convex, it can be decomposed into a convex and a concave part. Thus the problem can be solved using the constrained concave-convex procedure (CCCP) [19, 23].

### 3.1 Optimization using the CCCP algorithm

The CCCP solves the optimization problem using an iterative minimization process. At each round $r$, given an initial $\boldsymbol{w}^{(r)}$, the CCCP replaces the concave part of the objective function with its first-order Taylor expansion at $\boldsymbol{w}^{(r)}$, and then sets $\boldsymbol{w}^{(r+1)}$ to the solution of the relaxed optimization problem. When this function is non-smooth, such as $\max_{\boldsymbol{z} \in \mathcal{Z}_i} \mathbf{F}(\mathcal{X}_i, \boldsymbol{z}; \boldsymbol{w})$ in our formulation, the gradient in the Taylor expansion must be replaced by the subgradient[2]. Thus, at the $r$-th round, the

CCCP replaces $\max_{\boldsymbol{z} \in \mathcal{Z}_i} \mathbf{F}(\mathcal{X}_i, \boldsymbol{z}; \boldsymbol{w})$ in the loss function by

$$\max_{\boldsymbol{z} \in \mathcal{Z}_i} \mathbf{F}(\mathcal{X}_i, \boldsymbol{z}; \boldsymbol{w}^{(r)}) + (\boldsymbol{w} - \boldsymbol{w}^{(r)}) \cdot \partial \left( \max_{\boldsymbol{z} \in \mathcal{Z}_i} \mathbf{F}(\mathcal{X}_i, \boldsymbol{z}; \boldsymbol{w}) \right) . \tag{6}$$

The subgradient of a point-wise maximum function $g(\boldsymbol{x}) = \max_i g_i(\boldsymbol{x})$ is the convex hull of the union of subdifferentials of the subset of the functions $g_i(\boldsymbol{x})$ which equal $g(\boldsymbol{x})$ [4]. Defining by $\mathcal{C}_i^{(r)} = \{ \boldsymbol{z} \in \mathcal{Z}_i : \mathbf{F}(\mathcal{X}_i, \boldsymbol{z}; \boldsymbol{w}^{(r)}) = \max_{\boldsymbol{z}' \in \mathcal{Z}_i} \mathbf{F}(\mathcal{X}_i, \boldsymbol{z}'; \boldsymbol{w}^{(r)}) \}$, the subgradient of the function $\max_{\boldsymbol{z} \in \mathcal{Z}_i} \mathbf{F}(\mathcal{X}_i, \boldsymbol{z}; \boldsymbol{w})$ equals to $\sum_l \alpha_{i,l}^{(r)} \partial \mathbf{F}(\mathcal{X}_i, \boldsymbol{z}_{i,l}; \boldsymbol{w}) = \sum_l \alpha_{i,l}^{(r)} \Phi(\mathcal{X}_i, \boldsymbol{z}_{i,l})$, with $\sum_l \alpha_{i,l}^{(r)} = 1$, and $\alpha_{i,l}^{(r)} \geq 0$ if $\boldsymbol{z}_{i,l} \in \mathcal{C}_i^{(r)}$ and $\alpha_{i,l} = 0$ otherwise. Hence we have

$$\sum_l \alpha_{i,l}^{(r)} \boldsymbol{w}^{(r)} \cdot \Phi(\mathcal{X}_i, \boldsymbol{z}_{i,l}) = \max_{\boldsymbol{z} \in \mathcal{Z}_i} \left( \boldsymbol{w}^{(r)} \cdot \Phi(\mathcal{X}_i, \boldsymbol{z}) \right) \sum_{l: \boldsymbol{z}_{i,l} \in \mathcal{C}_i^{(r)}} \alpha_{i,l}^{(r)} = \max_{\boldsymbol{z} \in \mathcal{Z}_i} \left( \boldsymbol{w}^{(r)} \cdot \Phi(\mathcal{X}_i, \boldsymbol{z}) \right) .$$

We are free to choose the values of the $\alpha_{i,l}^{(r)}$ in the convex hull, here we choose to set $\alpha_{i,l}^{(r)} = 1/|\mathcal{C}_i^{(r)}|$ for $\forall \boldsymbol{z}_{i,l} \in \mathcal{C}_i^{(r)}$. Using (6) the new loss function becomes

$$\ell_{cccp}^{(r)}(\mathcal{X}_i, \mathcal{Z}_i; \boldsymbol{w}) = \left| \max_{\bar{\boldsymbol{z}} \notin \mathcal{Z}_i} \left( \ell_{\Delta}^A(\bar{\boldsymbol{z}}, \mathcal{Z}_i) + \boldsymbol{w} \cdot \Phi(\mathcal{X}_i, \bar{\boldsymbol{z}}) \right) - \boldsymbol{w} \cdot \frac{1}{|\mathcal{C}_i^{(r)}|} \sum_{\boldsymbol{z} \in \mathcal{C}_i^{(r)}} \Phi(\mathcal{X}_i, \boldsymbol{z}) \right|_+ , \tag{7}$$

Replacing the non-convex loss $\ell_{max}$ in (5) with (7), the relaxed convex optimization program at $r$-th round of the CCCP is

$$\min_{\boldsymbol{w}} \frac{\lambda}{2} \|\boldsymbol{w}\|_2^2 + \frac{1}{N} \sum_{i=1}^N \ell_{cccp}^{(r)}(\mathcal{X}_i, \mathcal{Z}_i; \boldsymbol{w}) \tag{8}$$

With our choice of $\alpha_{i,l}^{(r)}$, in the first round of the CCCP when $\boldsymbol{w}$ is initialized at $\mathbf{0}$, the second $\max(\cdot)$ in (3) is approximated by the average over all the labeling vectors. The CCCP algorithm is guaranteed to decrease the objective function and it converges to a local minimum solution of (5) [23].

## 3.2 Solve the convex optimization problem using the Pegasos framework

In order to solve the relaxed convex optimization problem (8) efficiently at each round of the CCCP, we have designed a stochastic subgradient descent algorithm, using the Pegasos framework developed in [18]. At each step the algorithm takes $K$ random samples from the training set and calculates an estimate of the subgradient of the objective function using these samples. Then it performs a subgradient descent step with decreasing learning rate, followed by a projection of the solution into the space where the optimal solution lives. An upper bound on the radius of the ball in which the optimal hyperplane lives can be calculated by considering that

$$\frac{\lambda}{2} \|\boldsymbol{w}^*\|_2^2 \leq \min_{\boldsymbol{w}} \frac{\lambda}{2} \|\boldsymbol{w}\|_2^2 + \frac{1}{N} \sum_{i=1}^N \ell_{cccp}^{(r)}(\mathcal{X}_i, \mathcal{Z}_i; \boldsymbol{w}) \leq B$$

where $\boldsymbol{w}^*$ is the optimal solution of (8), and $B = \max_i(\ell_{cccp}^{(r)}(X_i, Z_i; \mathbf{0}))$. If we use $\Delta(z_m, y_m) = \mathbf{1}(z_m \neq y_m)$ in (7), B equals the maximum number of instances in the bags. The details of the Pegasos algorithm for solving (8) are given in Algorithm 2. Using the theorems in [18] it is easy to show that after $\widetilde{\mathcal{O}}(1/(\lambda \varepsilon))$ iterations Algorithm 2 converges in expectation to a solution of accuracy $\varepsilon$.

**Efficient implementation.** Note that even if we solve the problem in the primal, we can still use nonlinear kernels without computing the nonlinear mapping $\phi(\boldsymbol{x})$ explicitly. Since the implementation method is similar to the one described in [18, Section 4] for lack of space we omit the details.

Greedily searching for the most violating labeling vector $\hat{\boldsymbol{z}}_k$ in line 4 of Algorithm 2 can be computational expensive. Dynamic programming can be carried out to reduce the computational cost since the contribution of each instance is additive over different labels. Moreover, by looking into the structure of $\mathcal{Z}_i$, the computational time can be further reduced. In the general situation, the worst case complexity of searching the maximum of $\bar{\boldsymbol{z}} \notin \mathcal{Z}_i$ is $\mathcal{O}(\prod_{m=1}^{M_i} C_{i,m})$, where $C_{i,m}$ is the number of unique possible labels for $\boldsymbol{x}_{i,m}$ in $\mathcal{Z}_i$ (usually $C_{i,m} \ll L_i$). This complexity can be greatly reduced when there are special structures such as graphs and trees in the labeling set. See for example [20, Section 4] for a discussion on some specific problems and special cases.

---
**Algorithm 1** The CCCP algorithm for solving MMS
---
1: **initialize:** $\boldsymbol{w}^{(1)} = \mathbf{0}$
2: **repeat**
3:     Set $\mathcal{C}_i^{(r)} = \{\boldsymbol{z} \in \mathcal{Z}_i : \mathbf{F}(\mathcal{X}_i, \boldsymbol{z}; \boldsymbol{w}^{(r)}) = \max_{\boldsymbol{z}' \in \mathcal{Z}_i} \mathbf{F}(\mathcal{X}_i, \boldsymbol{z}'; \boldsymbol{w}^{(r)})\}$
4:     Set $\boldsymbol{w}^{(r+1)}$ as the solution of the convex optimization problem (8)
5: **until** convergence to a local minimum
6: **output:** $\boldsymbol{w}^{(r+1)}$
---

---
**Algorithm 2** Pegasos Algorithm for Solving Relaxed-MMS (8)
---
1: **Input:** $\boldsymbol{w}_0, \{\mathcal{X}_i, \boldsymbol{Z}_i, \mathcal{C}_i^{(r)}\}_{i=1}^N, \lambda, T, K, B$
2: **for** $t = 1, 2, \ldots, T$ **do**
3:     Draw at random $A_t \subseteq \{1, \ldots, N\}$, with $|A_t| = K$
4:     Compute $\hat{\boldsymbol{z}}_k = \arg\max_{\bar{\boldsymbol{z}} \notin \mathcal{Z}_k} \left( \ell_\Delta^A(\bar{\boldsymbol{z}}, \mathcal{Z}_k) + \boldsymbol{w}_t \cdot \Phi(\mathcal{X}_k, \bar{\boldsymbol{z}}) \right) \quad \forall k \in A_t$
5:     Set $A_t^+ = \{k \in A_t : \ell_{cccp}^{(r)}(\mathcal{X}_k, \mathcal{Z}_k; \boldsymbol{w}_t) > 0\}$
6:     Set $\boldsymbol{w}_{t+\frac{1}{2}} = (1 - \frac{1}{t})\boldsymbol{w}_t + \frac{1}{\lambda K t} \sum_{k \in A_t^+} \left( \sum_{\boldsymbol{z} \in \mathcal{C}_i^{(r)}} \Phi(\mathcal{X}_k, \boldsymbol{z})/|\mathcal{C}_i^{(r)}| - \Phi(\mathcal{X}_k, \hat{\boldsymbol{z}}_k) \right)$
7:     $\boldsymbol{w}_{t+1} = \min\left( 1, \sqrt{2B/\lambda}/\|\boldsymbol{w}_{t+\frac{1}{2}}\| \right) \boldsymbol{w}_{t+\frac{1}{2}}$
8: **end for**
9: **Output:** $\boldsymbol{w}_{T+1}$
---

## 4 Experiments

In order to evaluate the proposed algorithm, we first perform experiments on several artificial datasets created from standard machine learning databases. Finally, we test our algorithm on one of the examples motivating our study — learning a face recognition system from news images weakly annotated by their associated captions. We benchmark MMS against the following baselines:

- **SVM**: we train a fully-supervised SVM classifier using the ground-truth labels by considering every instance separately while ignoring the other candidate labels. Its performance can be considered as an upper bound for the performance using candidate labels. In all our experiments, we use the LIBLINEAR [11] package and test two different multiple-class extensions, the 1-vs-All method using L1-loss (1vA-SVM) and the method by Crammer and Singer [9] (MC-SVM).

- **CL-SVM**: the Candidate Labeling SVM (CL-SVM) is a naive approach which transforms the ambiguous labeled data into a standard supervised representation by treating all possible labels of each instance as true labels. Then it learns 1-vs-All SVM classifiers from the resulting dataset, where the negative examples are instances which do not have the corresponding label in their candidate labeling set. A similar baseline has been used in binary MIL literature [5].

- **MIML**: we also compared with two SVM-based MIML algorithms[3]: MIMLSVM [25] and M[3]MIML [24]. We train the MIML algorithms by treating the labels in $\mathcal{Z}_i$ as a label for the bag. During the test phase, we consider each instance separately and predict the labels as: $y = \arg\max_{y \in \mathcal{Y}} F_{\text{miml}}(\boldsymbol{x}, y)$, where $F_{\text{miml}}$ is the obtained classifier, and $F_{\text{miml}}(\boldsymbol{x}, y)$ can be interpreted as the confidence of the classifier in assigning the instance $\boldsymbol{x}$ to the class $y$. We would like to underline that although some of the experimental setups may favor our algorithm, we include the comparison between MMS and MIML algorithms because to the best of our knowledge it is the only existing principle framework for modeling instance bags with multiple labels. MIML algorithms may still have their own advantage in scenarios when no prior knowledge is available about the instances within a bag.

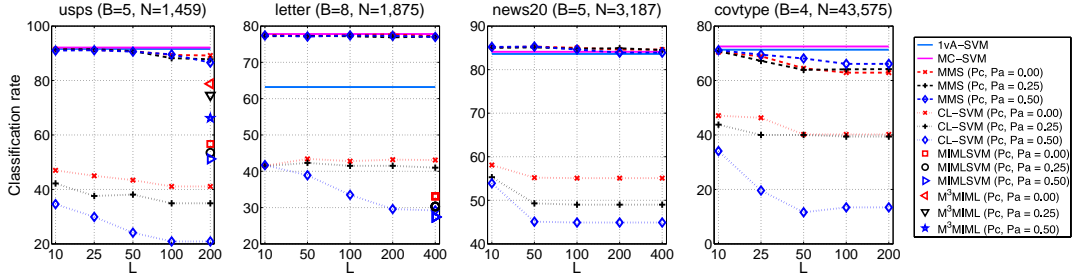

Figure 1: (**Best seen in colors**) Classification performance of different algorithms on artificial datasets.

We implemented our MMS algorithm in MATLAB[4], and used a value of the $1/N$ for the regularization parameter $\lambda$ in all our experiments. In (1) we used $\Delta(z_m, y_m) = \mathbf{1}(z_m \neq y_m)$. For a fair comparison, we used linear kernel for all the methods. The cost parameter for SVM algorithms is selected from the range $C \in \{0.1, 1, 10, 100, 1000\}$. The bias term is used in all the algorithms.

## 4.1 Experiments on artificial data

We create several artificial datasets using four widely used multi-class datasets (usps, letter, news20 and covtype) from the LIBSVM [6] website. The artificial training sets are created as follows: we first set at random pairs of classes as "correlated classes", and as "ambiguous classes", where the ambiguous classes can be different from the correlated classes. Following that, instances are grouped randomly into bags of fixed size $B$ with probability at least $P_c$ that two instances from correlated classes will appear in the same bag. Then $L$ ambiguous labeling vectors are created for each bag, by modifying a few elements of the correct labeling vector. The number of the modified element is randomly chosen from $\{1, \ldots, B\}$, and the new labels are chosen among a predefined ambiguous set. The ambiguous set is composed by the other correct labels from the same bag (except the true one) and a subset of the ambiguous pairs of all the correct labels from the bag. The probability of whether the ambiguous pair of a label is present equals $P_a$. For testing, we use the original test set, and each instance is considered separately.

Varying $P_c$, $P_a$, and $L$ we generate different dataset difficulty levels to evaluate the behaviour of the algorithms. For example, when $P_a > 0$, noisy labels are likely to be present in the labeling set. Meanwhile, $P_c$ controls the ambiguity within the same bags. If $P_c$ is large, instances from two correlated classes are likely to be grouped into the same bag, thus it becomes more difficult to distinguish between these two classes. The parameters $P_c$ and $P_a$ are chosen from $\{0, 0.25, 0.5\}$. For each difficulty level, we run three different training/test splits.

In figure 1, we plot the average classification accuracy. Several observations can be made: first, MMS achieves results close to the supervised SVM methods, and better than all other baselines. As MMS uses a similar multi-class loss as MC-SVM, it even outperforms 1vA-SVM when the loss has its advantage (e.g., on the 'letter' dataset). For the 'covtype' dataset, the performance gap between MMS and SVM is more visible. It may because 'covtype' has a class unbalance, where the two largest classes (among seven) dominate the whole dataset (more than 85% of the total number of samples). Second, the change on performance of MMS is small when the size of the candidate labeling set grows. Moreover, when correlated instances and extra noisy labels are present in the dataset, the baseline methods' performance drops by several percentages, while MMS is less affected. The CCCP algorithm usually converges in $3-5$ rounds, and the final performance is about $5\% - 40\%$ higher compared to the results obtained after the first round, especially when $L$ is large. This behavior also proves that approximating the second $\max(\cdot)$ function in the loss function (3) with the average over all the possible labeling vectors can lead to poor performance.

## 4.2 Applications to learning from images & captions

A huge amount of images with accompanying text captions are available on the web. This cheap source of information has been used, e.g., to name faces in images using captions [3, 13]. Thanks to the recent developments in the computer vision and natural language processing fields, faces in the images can be detected by a face detector and names in the captions can be identified using a language parser. The gathered data can then be used to train visual classifiers, without human's

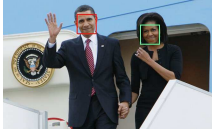

*President **Barack Obama** and first lady **Michelle Obama** wave from the steps of Air Force One as they arrive in Prague, Czech Republic.*

$$\begin{array}{cccccc} \boldsymbol{z}_1 & \boldsymbol{z}_2 & \boldsymbol{z}_3 & \boldsymbol{z}_4 & \boldsymbol{z}_5 & \boldsymbol{z}_6 \end{array}$$
$$\mathcal{Z}: \begin{bmatrix} \mathrm{n}_a & \mathrm{n}_a & \circ & \mathrm{n}_b & \circ & \mathrm{n}_b \\ \mathrm{n}_b & \circ & \mathrm{n}_b & \mathrm{n}_a & \mathrm{n}_a & \circ \end{bmatrix} \begin{array}{l} \leftarrow \mathrm{face}_a \\ \leftarrow \mathrm{face}_b \end{array}$$

Figure 2: (**Left**): An example image and its associated caption. There are two detected faces $\mathrm{face}_a$ and $\mathrm{face}_b$ and two names Barack Obama ($\mathrm{n}_a$) and Michelle Obama ($\mathrm{n}_b$) from the caption. (**Right**): The candidate labeling set for this image-captions pairs. The labeling vectors are generated using the following constrains: *i)*. a face in the image can either be assigned with a name from its caption, or it possibly corresponds to none of them (a *NULL* class, denoted as $\circ$); *ii)* a face can be assigned to at most one name; *iii)* a name can be assigned to at most a face. Differently from previous methods, we do not allow the labeling vector with all the faces assigned to the *NULL* class, because it would lead to the trivial solution with 0 loss by classifying every instance as *NULL*.

Table 1: Overall face recognition accuracy

| Dataset | 1vA-SVM | MC-SVM | CL-SVM | MIMLSVM | MMS |
|---------|---------|--------|--------|---------|-----|
| Yahoo! | $81.6\% \pm 0.6$ | $87.2\% \pm 0.3$ | $76.9\% \pm 0.2$ | $74.7\% \pm 0.9$ | $\mathbf{85.7\% \pm 0.5}$ |

effort in labeling the data. This task is difficult due to the so called "correspondence ambiguity" problem: there could be more than one face and name appearing in the image-caption pairs, and not all the names in the caption appear in the image, and vice versa. Nevertheless, this problem can be naturally formulated as a CLS problem. Since the names of the key persons in the image typically appear in the captions, combined with other common assumptions [3, 13], we can easily generate the candidate labeling sets (see Figure 2 for a practical example).

We conducted experiments on the Labeled Yahoo! News dataset[5] [3, 13]. The dataset is fully annotated for association of faces in the image with names in the caption, precomputed facial features were also available with the dataset. After preprocessing, the dataset contains 20071 images and 31147 faces. There are more than 10000 different names from the captions. We retain the 214 most frequent ones which occur at least 20 times, and treat the other names as *NULL*. The experiments are performed over 5 different permutations, sampling 80% images and captions as training set, and using the rest for testing. During splitting we also maintain the ratio between the number of samples from each class in the training and test set. For all algorithms, *NULL* names are considered as an additional class, except for MIML algorithms where unknown faces can be automatically considered as negative instances. The performance of the algorithms is measured by how many faces in the test set are correctly labeled with their name. Table 1 summarizes the results. Similar observations can also be made here: MMS achieves performance comparable to the fully-supervised SVM algorithms (4.1% higher than 1vA-SVM on Yahoo! data), while outperforming the other baselines for ambiguously labeled data.

## 5 Conclusion

In this paper, we introduce the "Candidate Labeling Set" problem where training samples contain multiple instances and a set of possible labeling vectors. We also propose a large margin formulation of the learning problem and an efficient algorithm for solving it. Although there are other similar frameworks, such as MIML, which also investigate learning from instance bags with multiple labels, our framework is different since it makes an explicit effort to label and to consider each instance in the bag during the learning process, and allows noisy labels in the training data. In particular, our framework provides a principled way to encode prior knowledge about relationships between instances and labels, and these constraints are explicitly taken into account into the loss function by the algorithm. The use of this framework does not have to be limited to data which is naturally grouped in multi-instance bags. It could be also possible to group separate instances into bags and solve the learning problem using MMS, when there are labeling constraints between these instances (e.g., a clustering problem with linkage constraints).

**Acknowledgments**   We thank the anonymous reviewers for their helpful comments. The Labeled Yahoo! News dataset were kindly provided by Matthieu Guillaumin and Jakob Verbeek. LJ was sponsored by the EU project DIRAC IST-027787 and FO was sponsored by the PASCAL2 NoE under EC grant no. 216886. LJ also acknowledges PASCAL2 Internal Visiting Programme for supporting traveling expense.

## Footnotes

[1]For simplicity we will omit the bias term here, it can be easily added by modifying the feature mapping.

[2]Given a function $g$, its subgradient $\partial g(\boldsymbol{x})$ at $\boldsymbol{x}$ satisfies: $\forall \boldsymbol{u}, g(\boldsymbol{u}) - g(\boldsymbol{x}) \geq \partial g(\boldsymbol{x}) \cdot (\boldsymbol{u} - \boldsymbol{x})$. The set of all subgradients of $g$ at $\boldsymbol{x}$ is called the subdifferential of $g$ at $\boldsymbol{x}$.

[3]We used the original implementation at http://lamda.nju.edu.cn/data.ashx#code. We did not compare against MIMLBOOST [25], because it does not scale to all the experiments we conducted. Besides, MIMLSVM [25] does not scale to data with high dimensional feature vectors (e.g., news20 which has a 62,061-dimensions features). Running the MATLAB implementation of M[3]MIML [24] on problems with more than a few thousand samples is computational infeasible. Thus, we will only report results using this two baseline methods on small size problems, where they can be finished in a reasonable amount of time.

[4]Code available at `http://dogma.sourceforge.net/`

[5]Dataset available at `http://lear.inrialpes.fr/data/`

# References

[1] S. Andrews, I. Tsochantaridis, and T. Hofmann. Support vector machines for multiple-instance learning. In *Proc. NIPS*, 2003.

[2] K. Barnard, P. Duygulu, D. Forsyth, N. de Freitas, D. Blei, and M. Jordan. Matching words and pictures. *JMLR*, 3:1107–1135, 2003.

[3] T. Berg, A. Berg, J. Edwards, and D. Forsyth. Who's in the picture? In *Proc. NIPS*, 2004.

[4] D. P. Bertsekas. *Convex Analysis and Optimization*. Athena Scientific, 2003.

[5] R. C. Bunescu and R. J. Mooney. Multiple instance learning for sparse positive bags. In *Proc. ICML*, 2007.

[6] C. C. Chang and C. J. Lin. *LIBSVM: A Library for Support Vector Machines*, 2001. Software available at `http://www.csie.ntu.edu.tw/~cjlin/libsvm`.

[7] O. Chapelle, A. Zien, and B. Schölkopf (Eds.). *Semi-supervised Learning*. MIT Press, 2006.

[8] T. Cour, B. Sapp, C. Jordan, and B. Taskar. Learning from ambiguously labeled images. In *Proc. CVPR*, 2009.

[9] K. Crammer and Y. Singer. On the algorithmic implementation of multiclass kernel-based vector machines. *JMLR*, 2:265–292, 2001.

[10] T. G. Dietterich, R. H. Lathrop, T. Lozano-Perez, and A. Pharmaceutical. Solving the multiple-instance problem with axis-parallel rectangles. *Artificial Intelligence*, 39:31–71, 1997.

[11] R.-E. Fan, K.-W. Chang, C.-J. Lin, S. S. Keerthi, and S. Sundarajan. LIBLINEAR: A library for large linear classification. *JMLR*, 9:1871–1874, 2008.

[12] Y. Grandvalet. Logistic regression for partial labels. In *Proc. IPMU*, 2002.

[13] M. Guillaumin, J. Verbeek, and C. Schmid. Multiple instance metric learning from automatically labeled bags of faces. In *Proc. ECCV*, 2010.

[14] A. Gupta and L. Davis. Beyond nouns: Exploiting prepositions and comparative adjectives for learning visual classifiers. In *Proc. ECCV*, 2008.

[15] E. Hüllermeier and J. Beringe. Learning from ambiguously labelled example. *Intelligent Data Analysis*, 10:419–439, 2006.

[16] L. Jie, B. Caputo, and V. Ferrari. Who's doing what: Joint modeling of names and verbs for simultaneous face and pose annotation. In *Proc. NIPS*, 2009.

[17] R. Jin and Z. Ghahramani. Learning with multiple labels. In *Proc. NIPS*, 2002.

[18] S. Shalev-Shwartz, Y. Singer, and N. Srebro. Pegasos: Primal Estimated sub-GrAdient SOlver for SVM. In *Proc. ICML*, 2007.

[19] A. J. Smola, S. V. N. Vishwanathan, and T. Hofmann. Kernel methods for missing variables. In *Proc. AISTAT*, 2005.

[20] I. Tsochantaridis, T. Joachims, T. Hofmann, and Y. Altun. Large margin methods for structured and interdependent output variables. *JMLR*, 6:1453–1484, 2005.

[21] E.P Xing, A.Y. Ng, M.I. Jordan, and S. Russell. Distance metric learning with application to clustering with side-information. In *Proc. NIPS*, 2002.

[22] C.-N. Yu and T. Joachims. Learning structural svms with latent variables. In *Proc. ICML*, 2009.

[23] A. Yuille and A. Rangarajan. The concave-convex procedure. *Neural Computation*, 15:915–936, 2003.

[24] M.-L. Zhang and Z.-H. Zhou. $M^3$MIML: A maximum margin method for multi-instance multi-label learning. In *Proc. ICDM*, 2008.

[25] Z.-H. Zhou and M.-L. Zhang. Multi-instance multi-label learning with application to scene classification. In *Proc. NIPS*, 2006.

[26] X. Zhu. Semi-supervised learning literature survey. Technical Report 1530, Computer Sciences, University of Wisconsin-Madison, 2005.

